# Learning Stochastic Perceptrons Under k-Blocking Distributions

**Mario Marchand**
Ottawa-Carleton Institute for Physics
University of Ottawa
Ottawa, Ont., Canada K1N 6N5
mario@physics.uottawa.ca

**Saeed Hadjifaradji**
Ottawa-Carleton Institute for Physics
University of Ottawa
Ottawa, Ont., Canada K1N 6N5
saeed@physics.uottawa.ca

## Abstract

We present a statistical method that PAC learns the class of stochastic perceptrons with arbitrary monotonic activation function and weights $w_i \in \{-1, 0, +1\}$ when the probability distribution that generates the input examples is member of a family that we call *k-blocking distributions*. Such distributions represent an important step beyond the case where each input variable is statistically independent since the 2k-blocking family contains all the Markov distributions of order k. By stochastic perceptron we mean a perceptron which, upon presentation of input vector **x**, outputs 1 with probability $f(\sum_i w_i x_i - \theta)$. Because the same algorithm works for any monotonic (nondecreasing or nonincreasing) activation function $f$ on Boolean domain, it handles the well studied cases of sigmoïds and the "usual" radial basis functions.

## 1 INTRODUCTION

Within recent years, the field of computational learning theory has emerged to provide a rigorous framework for the design and analysis of learning algorithms. A central notion in this framework, known as the "Probably Approximatively Correct" (PAC) learning criterion (Valiant, 1984), has recently been extended (Hassler, 1992) to analyze the learnability of *probabilistic* concepts (Kearns and Schapire, 1994; Schapire, 1992). Such concepts, which are stochastic rules that give the probability that input example **x** is classified as being positive, are natural probabilistic

extensions of the deterministic concepts originally studied by Valiant (1984).

Motivated by the stochastic nature of many "real-world" learning problems and by the indisputable fact that biological neurons are probabilistic devices, some preliminary studies about the PAC learnability of simple probabilistic neural concepts have been reported recently (Golea and Marchand, 1993; Golea and Marchand, 1994). However, the probabilistic behaviors considered in these studies are quite specific and clearly need to be extended. Indeed, only classification noise superimposed on a deterministic signum function was considered in Golea and Marchand (1993). The probabilistic network, analyzed in Golea and Marchand (1994), consists of a linear superposition of signum functions and is thus solvable as a (simple) case of linear regression. What is clearly needed is the extension to the non-linear cases of sigmoïds and radial basis functions. Another criticism about Golea and Marchand (1993, 1994) is the fact that their learnability results was established only for distributions where each input variable is statistically independent from all the others (sometimes called product distributions). In fact, very few positive learning results for non-trivial p-concepts classes are known to hold for larger classes of distributions. Therefore, in an effort to find algorithms that will work in practice, we introduce in this paper a new family of distributions that we call *k-blocking*. As we will argue, this family has the dual advantage of avoiding malicious and unnatural distributions that are prone to render simple concept classes unlearnable (Lin and Vitter, 1991) and of being likely to contain several distributions found in practice.

Our main contribution is to present a simple statistical method that PAC learns (in polynomial time) the class of stochastic perceptrons with monotonic (but otherwise arbitrary) activation functions and weights $w_i \in \{-1, 0, +1\}$ when the input examples are generated according to any distribution member of the k-blocking family. Due to space constraints, only a sketch of the proofs is presented here.

## 2   DEFINITIONS

The instance (input) space, $\mathcal{I}^n$, is the Boolean domain $\{-1, +1\}^n$. The set of all input variables is denoted by $X$. Each input example $\mathbf{x}$ is generated according to some unknown distribution $D$ on $\mathcal{I}^n$. We will often use $p_D(\mathbf{x})$, or simply $p(\mathbf{x})$, to denote the probability of observing the vector value $\mathbf{x}$ under distribution $D$. If $U$ and $V$ are two disjoint subsets of $X$, $\mathbf{x}_U$ and $\mathbf{x}_V$ will denote the restriction (or projection) of $\mathbf{x}$ over the variables of $U$ and $V$ respectively and $p_D(\mathbf{x}_U|\mathbf{x}_V)$ will denote the probability, under distribution $D$, of observing the vector value $\mathbf{x}_U$ (for the variables in $U$) given that the variables in $V$ are set to the vector value $\mathbf{x}_V$.

Following Kearns and Schapire (1994), a *probabilistic concept* (p-concept) is a map $c : \mathcal{I}^n \to [0, 1]$ for which $c(\mathbf{x})$ represents the probability that example $\mathbf{x}$ is classified as positive. More precisely, upon presentation of input $\mathbf{x}$, an output of $\sigma = 1$ is generated (by an unknown target p-concept) with probability $c(\mathbf{x})$ and an output of $\sigma = 0$ is generated with probability $1 - c(\mathbf{x})$.

A *stochastic perceptron* is a p-concept parameterized by a vector of $n$ weights $w_i$ and a *activation function* $f(\cdot)$ such that, the probability that input example $\mathbf{x}$ is

classified as positive is given by

$$\Pr\left(\sigma = 1 \,|\mathbf{x}\right) \;=\; f\left(\sum_{i=1}^{n} w_i x_i\right) \;.$$     (1)

We consider the case of a non-linear function $f(\cdot)$ since the linear case can be solved by a standard least square approximation like the one performed by Kearns in Schapire (1994) for linear sums of basis functions. We restrict ourselves to the case where $f(\cdot)$ is *monotonic i.e.* either nondecreasing or nonincreasing. But since any nonincreasing $f(\cdot)$ combined with a weight vector $\mathbf{w}$ can always be represented by a nondecreasing $f(\cdot)$ combined with a weight vector $-\mathbf{w}$, we can assume without loss of generality that the target stochastic perceptron has a nondecreasing $f(\cdot)$. Hence, we allow any sigmoïd-type of activation function (with arbitrary threshold). Also, since our instance space $\mathcal{I}^n$ is on a $n$-sphere, eq. 1 also include any nonincreasing radial basis function of the type $\phi(z^2)$ where $z = |\mathbf{x} - \mathbf{w}|$ and $\mathbf{w}$ is interpreted as the "center" of $\phi$. The only significant restriction is on the weights where we allow only for $w_i \in \{-1, 0, +1\}$.

As usual, the goal of the learner is to return an *hypothesis* $h$ which is a good approximation of the target p-concept $c$. But, in contrast with *decision rule learning* which attempts to "filter out" the noisy behavior by returning a deterministic hypothesis, the learner will attempt the harder (and more useful) task of modeling the target p-concept by returning a p-concept hypothesis. As a measure of error between the target and the hypothesis p-concepts we adopt the *variation distance* $d_v(\cdot, \cdot)$ defined as:

$$err(h, c) \;=\; d_v(h, c) \stackrel{\text{def}}{=} \sum_{\mathbf{x}} p_D(\mathbf{x}) \, |h(\mathbf{x}) - c(\mathbf{x})|$$     (2)

Where the summation is over all the $2^n$ possible values of $\mathbf{x}$. Hence, the same $D$ is used for both training and testing. The following formulation of the PAC criterion (Valiant, 1984; Hassler, 1992) will be sufficient for our purpose.

**Definition 1** *Algorithm A is said to PAC learn the class C of p-concepts by using the hypothesis class H (of p-concepts) under a family $\mathcal{D}$ of distributions on instance space $\mathcal{I}^n$, iff for any $c \in C$, any $D \in \mathcal{D}$, any $0 < \epsilon, \delta < 1$, algorithm A returns in a time polynomial in $(1/\epsilon, 1/\delta, n)$, an hypothesis $h \in H$ such that with probability at least $1 - \delta$, $err(h, c) < \epsilon$.*

## 3   K-BLOCKING DISTRIBUTIONS

To learn the class of stochastic perceptrons, the algorithm will try to discover each weight $w_i$ that connects to input variable $x_i$ by estimating how the probability of observing a positive output ($\sigma = 1$) is affected by "hard-wiring" variable $x_i$ to some fixed value. This should clearly give some information about $w_i$ when $x_i$ is statistically independent from all the other variables as was the case for Golea and Marchand (1993) and Schapire (1992). However, if the input variables are correlated, then the process of fixing variable $x_i$ will carry over neighboring variables which in turn will affect other variables until all the variables are perturbed (even in the simplest case of a first order Markov chain). The information about $w_i$ will

then be smeared by all the other weights. Therefore, to obtain information only on $w_i$, we need to break this "chain reaction" by fixing some other variables. The notion of *blocking sets* serves this purpose.

Loosely speaking, a set of variables is said to be a blocking set[1] for variable $x_i$ if the distribution on all the remaining variables is unaffected by the setting of $x_i$ whenever all the variables of the blocking set are set to a fixed value. More precisely, we have:

**Definition 2** *Let $B$ be a subset of $X$ and let $U = X - (B \cup \{x_i\})$. Let $\mathbf{x}_B$ and $\mathbf{x}_U$ be the restriction of $\mathbf{x}$ on $B$ and $U$ respectively and let $\mathbf{b}$ be an assignment for $\mathbf{x}_B$. Then $B$ is said to be a blocking set for variable $x_i$ (with respect to $D$), iff:*

$$p_D(\mathbf{x}_U | \mathbf{x}_B = \mathbf{b}, x_i = +1) = p_D(\mathbf{x}_U | \mathbf{x}_B = \mathbf{b}, x_i = -1) \quad \text{for all } \mathbf{b} \text{ and } \mathbf{x}_U$$

*In addition, if $B$ is not anymore a blocking set when we remove anyone of its variables, we then say that $B$ is a* minimal *blocking set for variable $x_i$.*

We thus adopt the following definition for the k-blocking family.

**Definition 3** *Distribution $D$ on $\mathcal{I}^n$ is said to be k-blocking iff $|B_i| \leq k$ for $i = 1, 2 \cdots n$ when each $B_i$ is a minimal blocking set for variable $x_i$.*

The k-blocking family is quite a large class of distributions. In fact we have the following property:

**Property 1** *All Markov distributions of kth order are members of the 2k-blocking family.*

**Proof:** By $k$th order Markov distributions, we mean distributions which can be exactly written as a Chow($k$) expansion (see Hoeffgen, 1993) for some permutation of the variables. We prove it here (by using standard techniques such as in Abend *et. al*, 1965) for first order Markov distributions, the generalization for $k > 1$ is straightforward. Recall that for Markov chain distributions we have: $p(x_j | x_{j-1}, \cdots x_1) = p(x_j | x_{j-1})$ for $1 < j \leq n$. Hence:

$$
\begin{aligned}
p(x_1 &\cdots x_{j-2}, x_{j+2} \cdots x_n | x_{j-1}, x_j, x_{j+1}) \\
&= p(x_1)p(x_2|x_1) \cdots p(x_j|x_{j-1})p(x_{j+1}|x_j) \cdots p(x_n|x_{n-1})/p(x_{j-1}, x_j, x_{j+1}) \\
&= p(x_1)p(x_2|x_1) \cdots p(x_{j-1}|x_{j-2})p(x_{j+2}|x_{j+1}) \cdots p(x_n|x_{n-1})/p(x_{j-1}) \\
&= p(x_1 \cdots x_{j-2}, x_{j+2} \cdots x_n | x_{j-1}, \overline{x_j}, x_{j+1})
\end{aligned}
$$

where $\overline{x_j}$ denotes the negation of $\mathbf{x}_j$. Thus, we see that Markov chain distributions are a special case of 2-blocking distributions: the blocking set of each variable consisting only of the two first-neighbor variables. $\square$.

The proposed algorithm for learning stochastic perceptrons needs to be provided with a blocking set (of at most $k$ variables) for each input variable. Hoeffgen (1993) has recently proven that Chow(1) and Chow($k > 1$) expansions are efficiently learnable; the latter under some restricted conditions. We can thus use these algorithms

to discover the blocking sets for such distributions. However, the efficient learnability of unrestricted Chow($k > 1$) expansions and larger classes of distributions, such as the k-blocking family, is still unknown. In fact, from the hardness results of Hoeffgen (1993), we can see that it is definitely very hard (perhaps NP-complete) to find the blocking sets if the learner has no information available other than the fact that the distribution is k-blocking. On the other hand, we can argue that the "natural" ordering of the variables present in many "real-world" situations is such that the blocking set of any given variable is among the neighboring variables. In vision for example, we expect that the setting of a pixel will directly affect only those located in it's neighborhood; the other pixels being affected only through this neighborhood. In such cases, the neighborhood of a variable "naturally" provides its blocking set.

## 4   LEARNING STOCHASTIC PERCEPTRONS

We first establish (the intuitive fact) that, without making much error, we can always consider that the target p-concept is defined only over the variables which are not almost always set to the same value.

**Lemma 1** *Let $V$ be a set of $v$ variables $x_i$ for which $\Pr(x_i = a_i) > 1 - \alpha$. Let $c$ be a p-concept and let $c'$ be the same p-concept as $c$ except that the reading of each variable $x_i \in V$ is replaced by the reading of the constant value $a_i$. Then $err(c', c) < v \cdot \alpha$.*

**Proof:** Let $\mathbf{a}$ be the vector obtained from the concatenation of all $a_i$s and let $\mathbf{x}_V$ be the vector obtained from $\mathbf{x}$ by keeping only the components $x_i$ which are in $V$. Then $err(c', c) \leq \Pr(\mathbf{x}_V \neq \mathbf{a}) \leq \sum_{i \in V} \Pr(x_i \neq a_i)$. $\square$

For a given set of blocking sets $\{B_i\}_{i=1}^n$, the algorithm will try to discover each weight $w_i$ by estimating the *blocked influence* of $x_i$ defined as:

$$\text{Binf}(x_i | \mathbf{b}_i) \stackrel{\text{def}}{=} \Pr(\sigma = 1 | \mathbf{x}_{B_i} = \mathbf{b}_i, x_i = +1) - \Pr(\sigma = 1 | \mathbf{x}_{B_i} = \mathbf{b}_i, x_i = -1)$$

where $\mathbf{x}_{B_i}$ denotes the restriction of $\mathbf{x}$ on the blocking set $B_i$ for variable $x_i$ and $\mathbf{b}_i$ is an assignment for $\mathbf{x}_{B_i}$. The following lemma ensures the learner that $\text{Binf}(x_i | \mathbf{b}_i)$ contains enough information about $w_i$.

**Lemma 2** *Let the target p-concept be a stochastic perceptron on $\mathcal{I}^n$ having a nondecreasing activation function and weights taken from $\{-1, 0, +1\}$. Then, for any assignment $\mathbf{b}_i$ for the variables in the blocking set $B_i$ of variable $x_i$, we have:*

$$\text{Binf}(x_i | \mathbf{b}_i) \begin{cases} \geq 0 & \text{if } w_i = +1 \\ = 0 & \text{if } w_i = 0 \\ \leq 0 & \text{if } w_i = -1 \end{cases} \tag{3}$$

**Proof sketch:** Let $U = X - (B_i \cup \{x_i\})$, $s = \sum_{j \in U} w_j x_j$ and $\zeta = \sum_{k \in B_i} w_k b_k$. Let $p(s)$ denote the probability of observing $s$ (under $D$). Then $\text{Binf}(x_i | \mathbf{b}_i) = \sum_s p(s) \left[ f(s + \zeta + w_i) - f(s + \zeta - w_i) \right]$; from which we find the desired result for a nondecreasing $f(\cdot)$. $\square$

In principle, lemma 2 enables the learner to discover $w_i$ from $\mathrm{Binf}(x_i|\mathbf{b}_i)$. The learner, however, has only access to its *empirical estimate* $\widehat{\mathrm{Binf}}(x_i|\mathbf{b}_i)$ from a finite sample. Hence, we will use Hoeffding's inequality (Hoeffding, 1963) to find the number of examples needed for a probability $p$ to be close to its empirical estimate $\hat{p}$ with high probability.

**Lemma 3 (Hoeffding, 1963)** *Let $Y_1,\ldots,Y_m$ be a sequence of $m$ independent Bernoulli trials, each succeeding with probability $p$. Let $\hat{p} = \sum_{i=1}^{m} Y_i/m$. Then:*

$$\Pr\left(|\hat{p} - p| > \epsilon\right) \;\leq\; 2\exp\left(-2m\epsilon^2\right)$$

Hence, by writing $\mathrm{Binf}(x_i|\mathbf{b}_i)$ in terms of (unconditional) probabilities that can be estimated from *all* the training examples, we find from lemma 3 that the number $m_0(\epsilon,\delta,n)$ of examples needed to have $|\widehat{\mathrm{Binf}}(x_i|\mathbf{b}_i) - \mathrm{Binf}(x_i|\mathbf{b}_i)| < \epsilon$ with probability at least $1 - \delta$ is given by:

$$m_0(\epsilon,\delta,n) \;\geq\; \frac{1}{2}\left(\frac{8}{\kappa\epsilon}\right)^2 \ln\left(\frac{4}{\delta}\right)$$

where $\kappa = \alpha^{k+1}$ is the lowest permissible value for $p_D(\mathbf{b}_i, x_i)$ (see lemma 1). So, if the minimal nonzero value for $|\mathrm{Binf}(x_i|\mathbf{b}_i)|$ is $\beta$, then the number of examples needed to find, with confidence at least $1 - \delta$, the exact value for $w_i$ among $\{-1,0,+1\}$ is such that we need to have: $\Pr(|\widehat{\mathrm{Binf}}(x_i|\mathbf{b}_i) - \mathrm{Binf}(x_i|\mathbf{b}_i)| < \beta/2) > 1 - \delta$. Thus, whenever $\beta$ is of $\Omega(e^{-n})$, we will need of $O(e^{2n})$ examples to find (with prob $> 1-\delta$) the value for $w_i$. So, in order to be able to PAC learn from a polynomial sample, we must arrange ourselves so that we do not need to worry about such low values for $|\mathrm{Binf}(x_i|\mathbf{b}_i)|$. We therefore consider the *maximum blocked influence* defined as:

$$\mathrm{Binf}(x_i) \stackrel{\text{def}}{=} \mathrm{Binf}(x_i|\mathbf{b}_i^*)$$

where $\mathbf{b}_i^*$ is the vector value for which $|\mathrm{Binf}(x_i|\mathbf{b}_i)|$ is the largest. We now show that the learner can ignore all variables $x_i$ for which $|\mathrm{Binf}(x_i)|$ is too small (without making much error).

**Lemma 4** *Let $c$ be a stochastic perceptron with nondecreasing activation function $f(\cdot)$ and weights taken from $\{-1,0,+1\}$. Let $V \subset X$ and let $c_V$ be the same stochastic perceptron as $c$ except that $w_i = 0$ for all $x_i \in V$ and its activation function is changed to $f(\cdot + \theta)$. Then, there always exists a value for $\theta$ such that:*

$$err(c_V, c) \;\leq\; \sum_{i \in V} |\mathrm{Binf}(x_i)|$$

**Proof sketch:** By induction on $|V|$. To first verify the lemma for $V = \{x_1\}$, let $\mathbf{b}$ be a vector of values for the setting of all $x_i \in B_1$ and let $\mathbf{x}_U$ be a vector of values for the setting of all $x_j \in U = X - (B_1 \cup \{x_1\})$. Let $s = \sum_{j \in U} w_j x_j$ and $\zeta = \sum_{j \in B_1} w_j x_j$, then for $\theta = w_1$, we have:

$$
\begin{aligned}
err(c_V, c) &= \sum_{\mathbf{x}_U}\sum_{\mathbf{b}} p_D(\mathbf{x}_U|\mathbf{b})p_D(\mathbf{b}|x_1 = -1)p_D(x_1 = -1) \\
&\quad \times |f(s + \zeta + w_1) - f(s + \zeta - w_1)| \quad \leq \quad |\mathrm{Binf}(x_1)|
\end{aligned}
$$

We now assume that the lemma holds for $V = \{x_1, x_2 \cdots x_k\}$ and prove it for $W = V \cup \{x_{k+1}\}$. Let $S = \{x_{k+1}\}$ and let $f(\cdot + \theta_W)$, $f(\cdot + \theta_V)$ and $f(\cdot + \theta_S)$ denote respectively the activation function for $c_W$, $c_V$ and $c_S$. By inspecting the expressions for $err(c_V, c)$ and $err(c_W, c_S)$, we can see that there always exist a value for $\theta_W \in \{\theta_V + w_{k+1}, \theta_V - w_{k+1}\}$ and $\theta_S \in \{w_{k+1}, -w_{k+1}\}$ such that $err(c_W, c_S) \leq err(c_V, c)$. And since $d_v(\cdot, \cdot)$ satisfies the triangle inequality, $err(c_W, c) \leq err(c_V, c) + |\text{Binf}(x_{k+1})|$. $\square$

After discovering the weights, the hypothesis p-concept $h$ returned by the learner will simply be the table look-up of the estimated probabilities of observing a positive classification given that $\sum_{i=1}^{n} w_i x_i = s$ for all $s$ values that are observed with sufficient probability (the hypothesis can output any value for the values of $s$ that are observed very rarely). We thus have the following learning algorithm for stochastic perceptrons.

### Algorithm LearnSP$(n, \epsilon, \delta, \{B_i\}_{i=1}^{n})$

1. Call $m = 128 \left(\frac{2n}{\epsilon}\right)^{2k+4} \ln \left(\frac{16n}{\delta}\right)$ training examples (where $k = max_i |B_i|$).

2. Compute $\hat{\text{Pr}}(x_i = +1)$ for each variable $x_i$. Neglect $x_i$ whenever we have $\hat{\text{Pr}}(x_i = +1) < \epsilon/(4n)$ or $\hat{\text{Pr}}(x_i = +1) > 1 - \epsilon/(4n)$.

3. For each variable $x_i$ and for each of its blocking vector value $\mathbf{b}_i$, compute $\hat{\text{Binf}}(x_i|\mathbf{b}_i)$. Let $\mathbf{b}_i^*$ be the value of $\mathbf{b}_i$ for which $|\hat{\text{Binf}}(x_i|\mathbf{b}_i)|$ is the largest. Let $\hat{\text{Binf}}(x_i) = \hat{\text{Binf}}(x_i|\mathbf{b}_i^*)$.

4. For each variable $x_i$:

   (a) Let $w_i = +1$ whenever $\hat{\text{Binf}}(x_i) > \epsilon/(4n)$.
   (b) Let $w_i = -1$ whenever $\hat{\text{Binf}}(x_i) < \epsilon/(4n)$.
   (c) Otherwise let $w_i = 0$

5. Compute $\hat{\text{Pr}}(\sum_{i=1}^{n} w_i x_i = s)$ for $s = -n, \ldots +n$.

6. Return the hypothesis p-concept $h$ formed by the table look-up:

$$h(\mathbf{x}) = h'(s) = \widehat{\text{Pr}} \left( \sigma = 1 \left| \sum_{i=1}^{n} w_i x_i = s \right. \right)$$

for all $s$ for which $\hat{\text{Pr}}(\sum_{i=1}^{n} w_i x_i = s) > \epsilon/(8n+8)$. For the other $s$ values, let $h'(s) = 0$ (or any other value).

**Theorem 1** *Algorithm* **LearnSP** *PAC learns the class of stochastic perceptrons on $\mathcal{I}^n$ with monotonic activation functions and weights $w_i \in \{-1, 0, +1\}$ under any k-blocking distribution (when a blocking set for each variable is known). The number of examples required is $m = 128 \left(\frac{2n}{\epsilon}\right)^{2k+4} \ln \left(\frac{16n}{\delta}\right)$ (and the time needed is $O(n \times m)$) for the returned hypothesis to make error at most $\epsilon$ with confidence at least $1 - \delta$.*

**Proof sketch:** From Hoeffding's inequality (lemma 3) we can show that this sample size is sufficient to ensure that:

- $\left| \hat{\Pr}(x_i = +1) - \Pr(x_i = +1) \right| < \epsilon/(4n)$ with confidence at least $1 - \delta/(4n)$

- $\left| \hat{\mathrm{Binf}}(x_i) - \mathrm{Binf}(x_i) \right| < \epsilon/(4n)$ with confidence at least $1 - \delta/(4n)$

- $\left| \hat{\Pr}(\sum_{i=1}^{n} w_i x_i = s) - \Pr(\sum_{i=1}^{n} w_i x_i = s) \right| < \epsilon^2/[64(n+1)]$ with confidence at least $1 - \delta/(4n+4)$

- $\left| \hat{\Pr}(\sigma = 1 | \sum_{i=1}^{n} w_i x_i = s) - \Pr(\sigma = 1 | \sum_{i=1}^{n} w_i x_i = s) \right| < \epsilon/4$ with confidence at least $1 - \delta/4$

From this and from lemma 1, 2 and 4, it follows that returned hypothesis will make error at most $\epsilon$ with confidence at least $1 - \delta$. □.

## Acknowledgments

We thank Mostefa Golea, Klaus-U. Hoeffgen and Stefan Poelt for useful comments and discussions about technical points. M. Marchand is supported by NSERC grant OGP0122405. Saeed Hadjifaradji is supported by the MCHE of Iran.

## Footnotes

[1]The wording "blocking set" was also used by Hancock & Mansour (*Proc. of COLT'91*, 179–183, Morgan Kaufmann Publ.) to denote a property of the target concept. In contrast, our definition of blocking set denotes a property of the input distribution only.

## References

Abend K., Hartley T.J. & Kanal L.N. (1965) "Classification of Binary Random Patterns", *IEEE Trans. Inform. Theory* vol. IT-11, 538–544.

Golea, M. & Marchand M. (1993) "On Learning Perceptrons with Binary Weights", *Neural Computation* vol. 5, 765–782.

Golea, M. & Marchand M. (1994) "On Learning Simple Deterministic and Probabilistic Neural Concepts", in Shawe-Talor J. , Anthony M. (eds.), *Computational Learning Theory: EuroCOLT'93*, Oxford University Press, pp. 47–60.

Haussler D. (1992) "Decision Theoritic Generalizations of the PAC Model for Neural Net and Other Learning Applications", *Information and Computation* vol. 100, 78–150.

Hoeffgen K.U. (1993) "On Learning and Robust Learning of Product Distributions", *Proceedings of the 6th ACM Conference on Computational Learning Theory*, ACM Press, 77–83.

Hoeffding W. (1963) "Probability inequalities for sums of bounded random variables", *Journal of the American Statistical Association*, vol. 58(301), 13–30.

Kearns M.J. and Schapire R.E. (1994) " Efficient Distribution-free Learning of Probabilistic Concepts", *Journal of Computer and System Sciences*, Vol. 48, pp. 464–497.

Lin J.H. & Vitter J.S. (1991) "Complexity Results on Learning by Neural Nets", *Machine Learning*, Vol. 6, 211-230.

Schapire R.E. (1992) *The Design and Analysis of Efficient Learning Algorithms*, Cambridge MA: MIT Press.

Valiant L.G. (1984) "A Theory of the Learnable", *Comm. ACM*, Vol. 27, 1134–1142.